# Dynamic Structure Super-Resolution

**Amos J Storkey**
Institute of Adaptive and Neural Computation
Division of Informatics and Institute of Astronomy
University of Edinburgh
5 Forrest Hill, Edinburgh UK
*a.storkey@ed.ac.uk*

## Abstract

The problem of super-resolution involves generating feasible higher resolution images, which are pleasing to the eye and realistic, from a given low resolution image. This might be attempted by using simple filters for smoothing out the high resolution blocks or through applications where substantial prior information is used to imply the textures and shapes which will occur in the images. In this paper we describe an approach which lies between the two extremes. It is a generic unsupervised method which is usable in all domains, but goes beyond simple smoothing methods in what it achieves. We use a dynamic tree-like architecture to model the high resolution data. Approximate conditioning on the low resolution image is achieved through a mean field approach.

## 1 Introduction

Good techniques for super-resolution are especially useful where physical limitations exist preventing higher resolution images from being obtained. For example, in astronomy where public presentation of images is of significant importance, super-resolution techniques have been suggested. Whenever dynamic image enlargement is needed, such as on some web pages, super-resolution techniques can be utilised. This paper focuses on the issue of how to increase the resolution of a *single* image using only prior information about images in general, and not relying on a specific training set or the use of multiple images.

The methods for achieving super-resolution are as varied as the applications. They range from simple use of Gaussian or preferably median filtering, to supervised learning methods based on learning image patches corresponding to low resolution regions from training data, and effectively sewing these patches together in a consistent manner. What method is appropriate depends on how easy it is to get suitable training data, how fast the method needs to be and so on. There is a demand for methods which are reasonably fast, which are generic in that they do not rely on having suitable training data, but which do better than standard linear filters or interpolation methods.

This paper describes an approach to resolution doubling which achieves this. The

method is structurally related to one layer of the dynamic tree model [9, 8, 1] except that it uses real valued variables.

## 2  Related work

Simple approaches to resolution enhancement have been around for some time. Gaussian and Wiener filters (and a host of other linear filters) have been used for smoothing the blockiness created by the low resolution image. Median filters tend to fare better, producing less blurry images. Interpolation methods such as cubic-spline interpolation tend to be the most common image enhancement approach.

In the super-resolution literature there are many papers which do not deal with the simple case of reconstruction based on a single image. Many authors are interested in reconstruction based on multiple slightly perturbed subsamples from an image [3, 2] . This is useful for photographic scanners for example. In a similar manner other authors utilise the information from a number of frames in a temporal sequence [4]. In other situations highly substantial prior information is given, such as the ground truth for a part of the image. Sometimes restrictions on the type of processing might be made in order to keep calculations in real time or deal with sequential transmission.

One important paper which deals specifically with the problem tackled here is by Freeman, Jones and Pasztor [5]. They follow a supervised approach, learning a low to high resolution patch model (or rather storing examples of such maps), and utilising a Markov random field for combining them and loopy propagation for inference. Later work [6] simplifies and improves on this approach. Earlier work tackling the same problem includes that of Schultz and Stevenson [7], which performed an MAP estimation using a Gibbs prior.

There are two primary difficulties with smoothing (eg Gaussian, Wiener, Median filters) or interpolation (bicubic, cubic spline) methods. First smoothing is indiscriminate. It occurs both within the gradual change in colour of the sky, say, as well as across the horizon, producing blurring problems. Second these approaches are inconsistent: subsampling the super-resolution image will not return the original low-resolution one. Hence we need a model which maintains consistency but also tries to ensure that smoothing does not occur across region boundaries (except as much is as needed for anti-aliasing).

## 3  The model

Here the high-resolution image is described by a series of very small patches with varying shapes. Pixel values within these patches can vary, but will have a common mean value. Pixel values across patches are independent. Apriori exactly where these patches should be is uncertain, and so the pixel to patch mapping is allowed to be a dynamic one.

The model is best represented by a belief network. It consists of three layers. The lowest layer consists of the visible low-resolution pixels. The intermediate layer is a high-resolution image ($4 \times 4$ the size of the low-resolution image). The top layer is a latent layer which is a little more than $2 \times 2$ the size of the low resolution image.

The latent variables are 'positioned' at the corners, centres and edge centres of the pixels of the low resolution image. The values of the pixel colour of the high resolution nodes are each a single sample from a Gaussian mixture (in colour space), where each mixture centre is given by the pixel colour of a particular parent latent

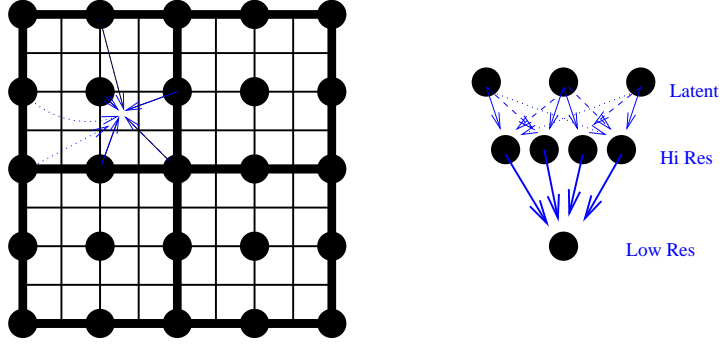

Figure 1: The three layers of the model. The small boxes in the left figure (64 of them) give the position of the high resolution pixels relative to the low resolution pixels (the 4 boxes with a thick outline). The positions of the latent variable nodes are given by the black circles. The colour of each high resolution pixel is generated from a mixture of Gaussians (right figure), each Gaussian centred at its latent parent pixel value. The closer the parent is, the higher the prior probability of being generated by that mixture is.

variable node. The prior mixing coefficients decay with distance in image space between the high-resolution node and the corresponding latent node.

Another way of viewing this is that a further indicator variable can be introduced which selects which mixture is responsible for a given high-resolution node. We say a high resolution node 'chooses' to connect to the parent that is responsible for it, with a connection probability given by the corresponding mixing coefficient. These connection probabilities can be specified in terms of positions (see figure 2).

The motivation for this model comes from the possibility of explaining away. In linear filtering methods each high-resolution node is determined by a fixed relationship to its neighbouring low-resolution nodes. Here if one of the latent variables provides an explanation for a high-resolution node which fits well with it neighbours to form the low-resolution data, then the posterior responsibility of the other latent nodes for that high-resolution pixel is reduced, and they are free to be used to model other nearby pixels. The high-resolution pixels corresponding to a visible node can be separated into two (or more) independent regions, corresponding to pixels on different sides of an edge (or edges). A different latent variable is responsible for each region. In other words each mixture component effectively corresponds to a small image patch which can vary in size depending on what pixels it is responsible for.

Let $\mathbf{v}_j \in L$ denote a latent variable at site $j$ in the latent space $L$. Let $\mathbf{x}_i \in S$ denote the value of pixel $i$ in high resolution image space $S$, and let $\mathbf{y}_k$ denote the value of the visible pixel $k$. Each of these is a 3-vector representing colour. Let $V$ denote the ordered set of all $\mathbf{v}_j$. Likewise $X$ denotes the ordered set of all $\mathbf{x}_i$ and $Y$ the set of all $\mathbf{y}_i$. In all the work described here a transformed colorspace of (gray, red-green, blue-yellow) is used. In other words the data is a linear transformation on the RGB colour values using the matrix

$$\begin{pmatrix} 0.66 & 1 & 0.5 \\ 0.66 & -1 & 0.5 \\ 0.66 & 0 & -1 \end{pmatrix}.$$

The remaining component is the connectivity (i.e. the indicator for the responsibility) between the high-resolution nodes and the nodes in the latent layer. Let $z_{ij}$

denote this connectivity with $z_{ij}$ an indicator variable taking value 1 when $v_j$ is a parent of $x_i$ in the belief network. Every high resolution pixel has one and only one parent in the latent layer. Let $Z$ denote the ordered set of all $z_{ij}$.

## 3.1 Distributions

A uniform distribution over the range of pixel values is presumed for the latent variables. The high resolution pixels are given by Gaussian distributions centred on the pixel values of the parental latent variable. This Gaussian is presumed independent in each pixel component. Finally the low resolution pixels are given by the average of the sixteen high resolution pixels covering the site of the low resolution pixel. This pixel value can also be subject to some additional Gaussian noise if necessary (zero noise is assumed in this paper).

It is presumed that each high resolution pixel is allowed to 'choose' its parent from the set of latent variables in an independent manner. A pixel has a higher probability of choosing a nearby parent than a far away one.

For this we use a Gaussian integral form so that :

$$P(Z) = \prod_{ij} p_{ij}^{z_{ij}} \text{ where } p_{ij} \propto \int_{B_i} d\mathbf{r} \exp\left(-\frac{(\mathbf{r}_j - \mathbf{r})^2}{2\Sigma}\right), \qquad (1)$$

where $\mathbf{r}$ is a position in the high resolution picture space, $\mathbf{r}_j$ is the position of the $j$th latent variable in the high resolution image space (where these are located at the corners of every second pixel in each direction as described above). The integral is over $B_i$ defined as the region in image space corresponding to pixel $x_i$. $\Sigma$ gives the width (squared) over which the probability decays. The larger $\Sigma$ the more possible parents with non-negligible probability. The connection probabilities can be illustrated by the picture in figure 2.

The equations for the other distributions are given here. First we have

$$P(X|Z,V) = \prod_{ijm} \frac{1}{(2\pi\Omega^m)^{1/2}} \exp\left(-z_{ij}\frac{(x_i^m - v_j^m)^2}{2\Omega^m}\right). \qquad (2)$$

where $\Omega^m$ is a variance which determines how much each pixel must be like its latent parent. Here the indicator $z_{ij}$ ensures the only contribution for each $i$ comes from the parent $j$ of $i$. Second

$$P(Y|X) = \prod_{km} \frac{1}{(2\pi\Lambda)^{1/2}} \exp\left(-\frac{(y_k^m - \frac{1}{d}\sum_{i\in Pa(k)} x_i^m)^2}{2\Lambda}\right) \qquad (3)$$

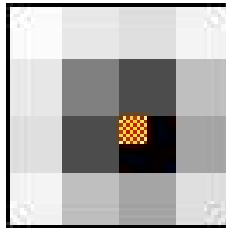

Figure 2: An illustration of the connection probabilities from a high resolution pixel in the position of the smaller checkered square to the latent variables centred at each of the larger squares. The probability is proportional to the intensity of the shading: darker is higher probability.

with $Pa(k)$ denoting the set of all the $d = 16$ high resolution pixels which go to make up the low resolution pixel $y_k$. In this work we let the variance $\Lambda \to 0$. $\Lambda$ determines the additive Gaussian noise which is in the low resolution image. Last, $P(V)$ is simply uniform over the whole of the possible values of $V$. Hence $P(V) = 1/C$ for $C$ the volume of $V$ space being considered.

## 3.2 Inference

The belief network defined above is not tree structured (rather it is a mixture of tree structures) and so we have to resort to approximation methods for inference. In this paper a variational approach is followed. The posterior distribution is approximated using a factorised distribution over the latent space and over the connectivity. Only in the high resolution space $X$ do we consider joint distributions: we use a joint Gaussian for all the nodes corresponding to one low resolution pixel. The full distribution can be written as $Q(Z, V, X) = Q(Z)Q(V)Q(X)$ where

$$Q(Z) = \prod_{ij} q_{ij}^{z_{ij}}, \qquad Q(V) = \prod_{jm} \frac{1}{(2\pi\Phi_j^m)^{1/2}} \exp\left(-\frac{(v_j^m - \nu_j^m)^2}{2(\Phi_j^m)}\right) \text{ and} \quad (4)$$

$$Q(X) = \prod_{km} \frac{(2\pi)^{-d/2}}{|\Psi_k^m|^{1/2}} \exp\left(-\frac{1}{2}[(\mathbf{x}^*)_k^m - (\boldsymbol{\mu}^*)_k^m]^T (\Psi_k^m)^{-1}[(\mathbf{x}^*)_k^m - (\boldsymbol{\mu}^*)_k^m]\right) \quad (5)$$

where $(\mathbf{x}^*)_k^m$ is the vector $(x_i^m | i \in Pa(k))$, the joint of all $d$ high resolution pixel values corresponding to a given low resolution pixel $k$ (for a given colour component $m$). Here $q_{ij}$, $\mu_i^m$, $\nu_j^m$, $\Phi_j^m$ and $\Psi_i^m$ are variational parameters to be optimised.

As usual, a local minima the KL divergence between the approximate distribution and the true posterior distribution is computed. This is equivalent to maximising the negative variational free energy (or variational log likelihood)

$$L(Q||P) = \left\langle \log \frac{Q(Z, V, X)}{P(Z, V, X, Y)} \right\rangle_{Q(Z,V,X)} \quad (6)$$

where $Y$ is given by the low resolution image. In this case we obtain

$$L(Q||P) = \langle \log Q(Z) - \log P(Z) \rangle_{Q(Z)} + \langle \log Q(V) - \log p(V) \rangle_{Q(V)}$$
$$+ \langle \log Q(X) \rangle_{Q(X)} - \langle \log P(X|Z, V) \rangle_{Q(X,Z,V)} - \langle \log P(Y|X) \rangle_{Q(Y,X)}. \quad (7)$$

Taking expectations and derivatives with respect to each of the parameters in the approximation gives a set of self-consistent mean field equations which we can solve by repeated iteration. Here for simplicity we only solve for $q_{ij}$ and for the means $\mu_i^m$ and $\nu_j^m$ which turn out to be independent of the variational variance parameters. We obtain

$$\nu_j^m = \frac{\sum_i q_{ij} x_i^m}{\sum_i q_{ij}} \text{ and } \mu_i^m = \rho_i^m + D_{c(i)} \text{ where } \rho_i^m = \sum_j q_{ij} v_i^m \quad (8)$$

where $c(i)$ is the child of $i$, i.e. the low level pixel which $i$ is part of. $D_k$ is a Lagrange multiplier, and is obtained through constraining the high level pixel values to average to the low level pixels:

$$\frac{1}{d} \sum_{i \in Pa(k)} \mu_i^m = y_k^m \implies D_k \equiv D_k^* = y_k^m - \frac{1}{d} \sum_{i \in Pa(k)} \rho_i^m \quad (9)$$

In the case where $\Lambda$ is non-zero, this constraint is softened and $D_k$ is given by $D_k = \Omega D_k^*/(\Omega + \Lambda)$. The update for the $q_{ij}$ is given by

$$q_{ij} \propto p_{ij} \exp\left(-\sum_m \frac{(x_i^m - v_k^m)^2}{2\Omega^m}\right) \quad (10)$$

where the constant of proportionality is given by normalisation: $\sum_j q_{ij} = 1$.

Optimising the KL divergence involves iterating these equations. For each $Q(Z)$ optimisation (10), equations (8a) and (8b) are iterated a number of times. Each optimisation loop is either done a preset number of times, or until a suitable convergence criterion is met. The former approach is generally used, as the basic criterion is a limit on the time available for the optimisation to be done.

## 4  Setting parameters

The prior variance parameters need to be set. The variance $\Lambda$ corresponds to the additive noise. If this is not known to be zero, then it will vary from image to image, and needs to be found for each image. This can be done using variational maximum likelihood, where $\Lambda$ is set to maximise the variational log likelihood. $\Sigma$ is presumed to be independent of the images presented, and is set by hand by visualising changes on a test set. The $\Omega_m$ might depend on the intensity levels in the image: very dark images will need a smaller value of $\Omega_1$ for example. However for simplicity $\Omega_m = \Omega$ is treated as global and set by hand. Because the primary criterion for optimal parameters is subjective, this is the most sensible approach, and is reasonable when there are only two parameters to determine. To optimise automatically based on the variational log likelihood is possible but does not produce as good results due to the complicated nature of a true prior or error-measure for images. For example, a highly elaborate texture offset by one pixel will give a large mean square error, but look almost identical, whereas a blurred version of the texture would give a smaller mean square error, but look much worse.

## 5  Implementation

The basic implementation involves setting the parameters, running the mean field optimisation and then looking at the result. The final result is a downsampled version of the $4 \times 4$ image to $2 \times 2$ size: the larger image is used to get reasonable anti-aliasing.

To initialise the mean field optimisation, $X$ is set equal to the bi-cubic interpolated image with added Gaussian noise. The $Q(Z)$ is initialised to $P(Z)$. Although in the examples here we used 25 optimisations $Q(Z)$, each of which involves 10 cycles through the mean field equations for $Q(X)$ and $Q(V)$, it is possible to get reasonable results with only three $Q(Z)$ optimisation cycles each doing 2 iterations through the mean field equations. In the runs shown here, $\Lambda$ is set to zero, the variance $\Omega$ is set to 0.008, and $\Sigma$ is set to 3.3.

## 6  Demonstrations and assessment

The method described in this paper is compared with a number of simple filtering and interpolation methods, and also with the methods of Freeman et al. The image from Freeman's website is used for comparison with that work (figure 3). Full colour comparisons for these and other images can be found at `http://www.anc.ed.ac.uk/~amos/superresolution.html`. First two linear filtering approaches are considered, the Wiener filter and a Gaussian filter. The third method is a median filter. Bi-cubic interpolation is also given.

Quantitative assessment of the quality of super-resolution results is always something of a difficulty because the basic criterion is human subjectivity. Even so we

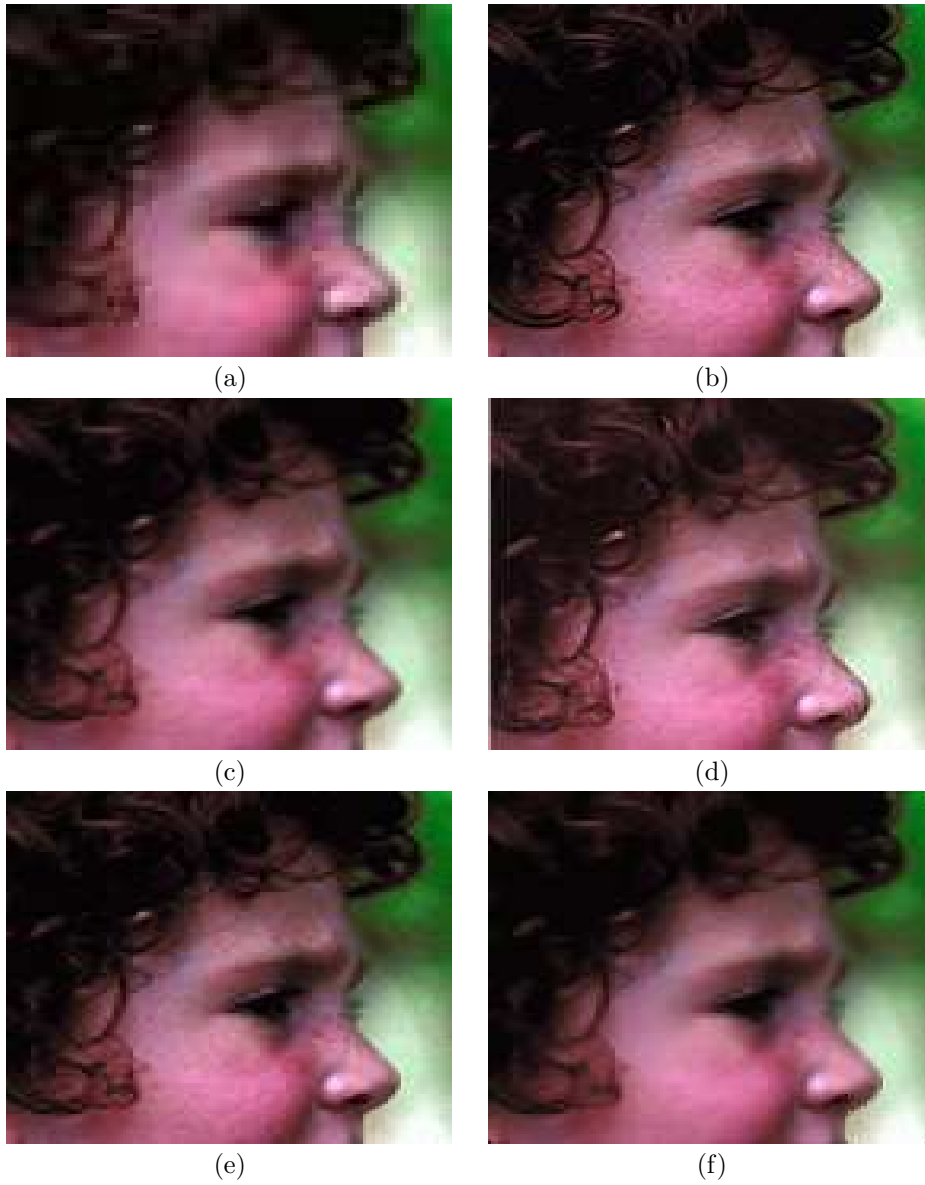

Figure 3: Comparison with approach of Freeman et al. (a) gives the 70x70 low resolution image, (b) the true image, (c) a bi-cubic interpolation (d) Freeman et al result (taken from website and downsampled), (e) dynamic structure super-resolution, (f) median filter.

compare the results of this approach with standard filtering methods using a root mean squared pixel error on a set of 8, 128 by 96 colour images, giving 0.0486, 0.0467, 0.0510 and 0.0452 for the original low resolution image, bicubic interpolation, the median filter and dynamic structure super-resolution respectively. Unfortunately the unavailability of code prevents representative calculations for the Freeman et al approach. Dynamic structure resolution requires approximately $30 - 60$ flops per $2 \times 2$ high resolution pixel per optimisation cycle, compared with, say, 16 flops for a linear filter, so it is more costly. Trials have been done working directly with $2 \times 2$ grids rather than with $4 \times 4$ and then averaging up. This is much faster and the results, though not quite as good, were still an improvement on the simpler methods.

Qualitatively, the results for dynamic structure super-resolution are significantly better than most standard filtering approaches. The texture is better represented because it maintains consistency, and the edges are sharper, although there is still some significant difference from the true image. The method of Freeman et al is perhaps comparable at this resolution, although it should be noted that their result has been downsampled here to half the size of their enhanced image. Their method can produce $4 \times 4$ the resolution of the original, and so this does not accurately represent the full power of their technique. Furthermore this image is representative of early results from their work. However their approach does require learning large numbers of patches from a training set. Fundamentally the dynamic structure super-resolution approach does a good job at resolution doubling without the need for representative training data. The edges are not blurred and much of the blockiness is removed.

Dynamic structure super-resolution provides a technique for resolution enhancement, and provides an interesting starting model which is different from the Markov random field approaches. Future directions could incorporate hierarchical frequency information at each node rather than just a single value.

## References

[1] N. J. Adams. *Dynamic Trees: A Hierarchical Probabilistic Approach to Image Modelling*. PhD thesis, Division of Informatics, University of Edinburgh, 5 Forrest Hill, Edinburgh, EH1 2QL, UK, 2001.

[2] S. Baker and T. Kanade. Limits on super-resolution and how to break them. In *Proceedings of CVPR 00*, pages 372–379, 2000.

[3] P. Cheeseman, B. Kanefsky, R. Kraft, and J. Stutz. Super-resolved surface reconstruction from multiple images. Technical Report FIA-94-12, NASA Ames, 1994.

[4] M. Elad and A. Feuer. Super-resolution reconstruction of image sequences. *IEEE Transactions on Pattern Analysis and Machine Intelligence*, 21(9):817–834, 1999.

[5] W. T. Freeman, T. R. Jones, and E. C. Pasztor. Markov networks for super-resolution. Technical Report TR-2000-08, MERL, 2000.

[6] W. T. Freeman, T. R. Jones, and E. C. Pasztor. Example-based super-resolution. *IEEE Computer Graphics and Applications*, 2002.

[7] R. R. Schultz and R. L. Stevenson. A Bayesian approach to image expansion for improved definition. *IEEE Transactions on Image Processing*, 3:233–242, 1994.

[8] A. J. Storkey. Dynamic trees: A structured variational method giving efficient propagation rules. In C. Boutilier and M. Goldszmidt, editors, *Uncertainty in Artificial Intelligence*, pages 566–573. Morgan Kauffmann, 2000.

[9] C. K. I. Williams and N. J. Adams. DTs: Dynamic trees. In M. J. Kearns, S. A. Solla, and D. A. Cohn, editors, *Advances in Neural Information Processing Systems 11*. MIT Press, 1999.
